# Monte Carlo POMDPs

**Sebastian Thrun**
School of Computer Science
Carnegie Mellon University
Pittsburgh, PA 15213

## Abstract

We present a Monte Carlo algorithm for learning to act in partially observable Markov decision processes (POMDPs) with real-valued state and action spaces. Our approach uses importance sampling for representing beliefs, and Monte Carlo approximation for belief propagation. A reinforcement learning algorithm, value iteration, is employed to learn value functions over belief states. Finally, a sample-based version of nearest neighbor is used to generalize across states. Initial empirical results suggest that our approach works well in practical applications.

## 1 Introduction

POMDPs address the problem of acting optimally in partially observable dynamic environment [6]. In POMDPs, a learner interacts with a stochastic environment whose state is only partially observable. Actions change the state of the environment and lead to numerical penalties/rewards, which may be observed with an unknown temporal delay. The learner's goal is to devise a policy for action selection that maximizes the reward. Obviously, the POMDP framework embraces a large range of practical problems.

Past work has predominately studied POMDPs in discrete worlds [1]. Discrete worlds have the advantage that distributions over states (so-called "belief states") can be represented exactly, using one parameter per state. The optimal value function (for finite planning horizon) has been shown to be convex and piecewise linear [10, 14], which makes it possible to derive exact solutions for discrete POMDPs.

Here we are interested in POMDPs with continuous state and action spaces, paying tribute to the fact that a large number of real-world problems are continuous in nature. In general, such POMDPs are not solvable exactly, and little is known about special cases that can be solved. This paper proposes an approximate approach, the MC-POMDP algorithm, which can accommodate real-valued spaces and models. The central idea is to use Monte Carlo sampling for belief representation and propagation. Reinforcement learning in belief space is employed to learn value functions, using a sample-based version of nearest neighbor for generalization. Empirical results illustrate that our approach finds to close-to-optimal solutions efficiently.

## 2 Monte Carlo POMDPs

### 2.1 Preliminaries

POMDPs address the problem of selection actions in stationary, partially observable, controllable Markov chains. To establish the basic vocabulary, let us define:

- *State.* At any point in time, the world is in a specific state, denoted by $x$.

- *Action.* The agent can execute actions, denoted $a$.
- *Observation.* Through its sensors, the agent can observe a (noisy) projection of the world's state. We use $o$ to denote observations.
- *Reward.* Additionally, the agent receives rewards/penalties, denoted $R \in \Re$. To simplify the notation, we assume that the *reward* is part of the observation. More specifically, we will use $R(o)$ to denote the function that "extracts" the reward from the observation.

Throughout this paper, we use the subscript $t$ to refer to a specific point in time (e.g., $s_t$ refers to the state at time $t$).

POMDPs are characterized by three probability distributions:

1. The *initial distribution*, $\pi(x) := Pr(x_0)$, specifies the initial distribution of states at time $t = 0$.
2. The *next state distribution*, $\mu(x' \mid a, x) := Pr(x_t = x' \mid a_{t-1} = a, x_{t-1} = x)$, describes the likelihood that action $a$, when executed at state $x$, leads to state $x'$.
3. The *perceptual distribution*, $\nu(o \mid x) := Pr(o_t = o \mid x_t = x)$, describes the likelihood of observing $o$ when the world is in state $x$.

A *history* is a sequence of states and observations. For simplicity, we assume that actions and observations are alternated. We use $d_t$ to denote the history leading up to time $t$:

$$d_t := \{o_t, a_{t-1}, o_{t-1}, a_{t-2}, \ldots, a_0, o_0\} \tag{1}$$

The fundamental problem in POMDPs is to devise a policy for action selection that maximizes reward. A *policy*, denoted

$$\sigma : d \longrightarrow a \tag{2}$$

is a mapping from histories to actions. Assuming that actions are chosen by a policy $\sigma$, each policy induces an expected cumulative (and possibly discounted by a *discount factor* $\gamma \leq 1$) reward, defined as

$$J^{\sigma} = \sum_{\tau=0}^{\infty} E\left[\gamma^{\tau} R(o_{\tau})\right] \tag{3}$$

Here $E[\ ]$ denotes the mathematical expectation. The POMDP problem is, thus, to find a policy $\sigma^*$ that maximizes $J^{\sigma}$, i.e.,

$$\sigma^* = \underset{\sigma}{\text{argmax}} \, J^{\sigma} \tag{4}$$

## 2.2 Belief States

To avoid the difficulty of learning a function with unbounded input (the history can be arbitrarily long), it is common practice to map histories into *belief states*, and learn a mapping from belief states to actions instead [10].

Formally, a *belief state* (denoted $\theta$) is a probability distribution over states conditioned on past actions and observations:

$$\theta_t = Pr(x_t \mid d_t) = Pr(x_t \mid o_t, a_{t-1}, \ldots, o_0) \tag{5}$$

Belief are computed incrementally, using knowledge of the POMDP's defining distributions $\pi$, $\mu$, and $\nu$. Initially

$$\theta_0 = \pi \tag{6}$$

For $t \geq 0$, we obtain

$$\theta_{t+1} = Pr(x_{t+1} \mid o_{t+1}, a_t, \ldots, o_0) \tag{7}$$

$$= \alpha \, Pr(o_{t+1} \mid x_{t+1}, \ldots, o_0) \, Pr(x_{t+1} \mid a_t, \ldots, o_0) \tag{8}$$

$$= \alpha \, Pr(o_{t+1} \mid x_{t+1}) \int Pr(x_{t+1} \mid a_t, \ldots, o_0, x_t) \, Pr(x_t \mid a_t, \ldots, o_0) \, dx_t \tag{9}$$

$$= \alpha \, Pr(o_{t+1} \mid x_{t+1}) \int Pr(x_{t+1} \mid a_t, x_t) \, \theta_t \, dx_t \tag{10}$$

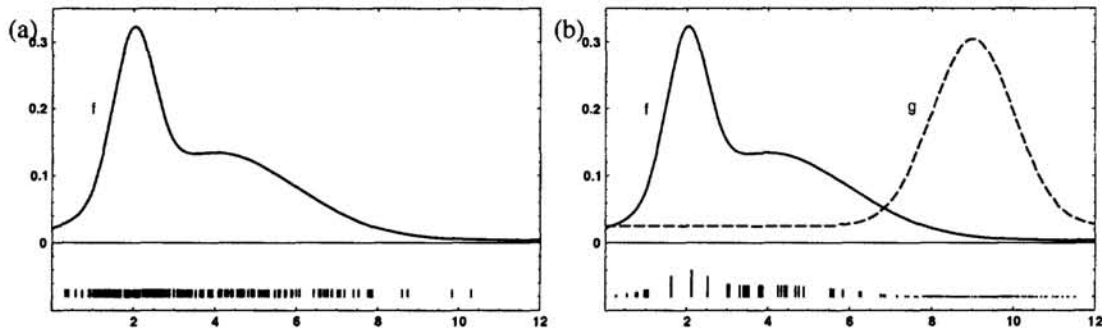

**Figure 1**: Sampling: (a) Likelihood-weighted sampling and (b) importance sampling. At the bottom of each graph, samples are shown that approximate the function $f$ shown at the top. The height of the samples illustrates their *importance factors*.

Here $\alpha$ denotes a constant normalizer. The derivations of (8) and (10) follow directly from the fact that the environment is a stationary Markov chain, for which future states and observations are conditionally independent from past ones given knowledge of the state. Equation (9) is obtained using the theorem of total probability.

Armed with the notion of belief states, the policy is now a mapping from belief states (instead of histories) to actions:

$$\sigma : \theta \longrightarrow a \tag{11}$$

The legitimacy of conditioning $a$ on $\theta$, instead of $d$, follows directly from the fact that the environment is Markov, which implies that $\theta$ is all one needs to know about the past to make optimal decisions.

### 2.3 Sample Representations

Thus far, we intentionally left open how belief states $\theta$ are *represented*. In prior work, state spaces have been discrete. In discrete worlds, beliefs can be represented by a collection of probabilities (one for each state), hence, beliefs can be represented exactly. Here were are interested in real-valued state spaces. In general, probability distributions over real-valued spaces possess infinitely many dimensions, hence cannot be represented on a digital computer.

The key idea is to represent belief states by *sets of (weighted) samples* drawn from the belief distribution. Figure 1 illustrates two popular schemes for sample-based approximation: *likelihood-weighted sampling*, in which samples (shown at the bottom of Figure 1a) are drawn directly from the target distribution (labeled $f$ in Figure 1a), and *importance sampling*, where samples are drawn from some other distribution, such as the curve labeled $g$ in Figure 1b. In the latter case, samples $x$ are annotated by a numerical importance factor

$$p(x) \quad = \quad \frac{f(x)}{g(x)} \tag{12}$$

to account for the difference in the sampling distribution, $g$, and the target distribution $f$ (the height of the bars in Figure 1b illustrates the importance factors). Importance sampling requires that $f > 0 \rightarrow g > 0$, which will be the case throughout this paper. Obviously, both sampling methods generate approximations only. Under mild assumptions, they converge to the target distribution at a rate of $\frac{1}{\sqrt{N}}$, with $N$ denoting the sample set size [16].

In the context of POMDPs, the use of sample-based representations gives rise to the following algorithm for approximate belief propagation (c.f., Equation (10)):

> **Algorithm particle_filter**$(\theta_t, a_t, o_{t+1})$:
>
>     $\theta_{t+1} = \emptyset$
>     *do N times:*
>         *draw random state* $x_t$ *from* $\theta_t$

> *sample $x_{t+1}$ according to $\mu(x_{t+1} \mid a_t, x_t)$*
> *set importance factor $p(x_{t+1}) = \nu(o_{t+1} \mid x_{t+1})$*
> *add $\langle x_{t+1}, p(x_{t+1}) \rangle$ to $\theta_{t+1}$*
> *normalize all $p(x_{t+1}) \in \theta_{t+1}$ so that $\sum p(x_{t+1}) = 1$*
> *return $\theta_{t+1}$*

This algorithm converges to (10) for arbitrary models $\mu$, $\nu$, and $\pi$ and arbitrary belief distributions $\theta$, defined over discrete, continuous, or mixed continuous-discrete state and action spaces. It has, with minor modifications, been proposed under names like *particle filters* [13], *condensation algorithm* [5], *survival of the fittest* [8], and, in the context of robotics, *Monte Carlo localization* [4].

### 2.4 Projection

In conventional planning, the result of applying an action $a_t$ at a state $x_t$ is a distribution $Pr(x_{t+1}, R_{t+1} \mid a_t, x_t)$ over states $x_{t+1}$ and rewards $R_{t+1}$ at the next time step. This operation is called *projection*. In POMDPs, the state $x_t$ is unknown. Instead, one has to compute the result of applying action $a_t$ to a belief state $\theta_t$. The result is a distribution $Pr(\theta_{t+1}, R_{t+1} \mid a_t, \theta_t)$ over belief states $\theta_{t+1}$ and rewards $R_{t+1}$. Since belief states themselves are distributions, the result of a projection in POMDPs is, technically, a distribution over distributions.

The projection algorithm is derived as follows. Using total probability, we obtain:

$$Pr(\theta_{t+1}, R_{t+1} \mid a_t, \theta_t) = Pr(\theta_{t+1}, R_{t+1} \mid a_t, d_t) \tag{13}$$

$$= \int \underbrace{Pr(\theta_{t+1}, R_{t+1} \mid o_{t+1}, a_t, d_t)}_{(*)} \underbrace{Pr(o_{t+1} \mid a_t, d_t)}_{(**)} \, do_{t+1} \tag{14}$$

The term $(*)$ has already been derived in the previous section (c.f., Equation (10)), under the observation that the reward $R_{t+1}$ is trivially computed from the observation $o_{t+1}$.

The second term, $(**)$, is obtained by integrating out the unknown variables, $x_{t+1}$ and $x_t$, and by once again exploiting the Markov property:

$$Pr(o_{t+1} \mid a_t, d_t) = \int Pr(o_{t+1} \mid x_{t+1}) \, Pr(x_{t+1} \mid a_t, d_t) \, dx_{t+1} \tag{15}$$

$$= \int Pr(o_{t+1} \mid x_{t+1}) \int Pr(x_{t+1} \mid x_t, a_t) \, Pr(x_t \mid d_t) \, dx_t \, dx_{t+1} \tag{16}$$

$$= \int \nu(o_{t+1} \mid x_{t+1}) \int \mu(x_{t+1} \mid x_t, a_t) \, \theta_t(x_t) \, dx_t \, dx_{t+1} \tag{17}$$

This leads to the following approximate algorithm for projecting belief state. In the spirit of this paper, our approach uses Monte Carlo integration instead of exact integration. It represents distributions (and distributions over distributions) by samples drawn from such distributions.

> **Algorithm particle_projection($\theta_t, a_t$):**
> $\Theta_t = \emptyset$
> *do $N$ times:*
>    *draw random state $x_t$ from $\theta_t$*
>    *sample a next state $x_{t+1}$ according to $\mu(x_{t+1} \mid a_t, x_t)$*
>    *sample an observation $o_{t+1}$ according to $\nu(o_{t+1} \mid x_{t+1})$*
>    *compute $\theta_{t+1}$ = particle_filter($\theta_t, a_t, o_{t+1}$)*
>    *add $\langle \theta_{t+1}, R(o_{t+1}) \rangle$ to $\Theta_t$*
> *return $\Theta_t$*

The result of this algorithm, $\Theta_t$, is a sample set of belief states $\theta_{t+1}$ and rewards $R_{t+1}$, drawn from the desired distribution $Pr(\theta_{t+1}, R_{t+1} \mid \theta_t, a_t)$. As $N \to \infty$, $\Theta_t$ converges with probability 1 to the true posterior [16].

## 2.5  Learning Value Functions

Following the rich literature on reinforcement learning [7, 15], our approach solves the POMDP problem by value iteration in belief space. More specifically, our approach recursively learns a value function $Q$ over belief states and action, by *backing up* values from subsequent belief states:

$$Q(\theta_t, a_t) \quad \longleftarrow \quad E\left[R(o_{t+1}) + \gamma \max_{\bar{a}} Q(\theta_{t+1}, \bar{a})\right] \tag{18}$$

Leaving open (for a moment) how $Q$ is represented, it is easy to be seen how the algorithm **particle_projection** can be applied to compute a Monte Carlo approximation of the right hand-side expression: Given a belief state $\theta_t$ and an action $a_t$, **particle_projection** computes a sample of $R(o_{t+1})$ and $\theta_{t+1}$, from which the expected value on the right hand side of (18) can be approximated.

It has been shown [2] that if both sides of (18) are equal, the *greedy* policy

$$\sigma^Q(\theta) \quad = \quad \operatorname*{argmax}_{\bar{a}} Q(\theta, \bar{a}) \tag{19}$$

is *optimal*, i.e., $\sigma^* = \sigma^Q$. Furthermore, it has been shown (for the discrete case!) that repetitive application of (18) leads to an optimal value function and, thus, to the optimal policy [17, 3].

Our approach essentially performs model-based reinforcement learning in belief space using approximate sample-based representations. This makes it possible to apply a rich bag of tricks found in the literature on MDPs. In our experiments below, we use on-line reinforcement learning with counter-based exploration and experience replay [9] to determine the order in which belief states are updated.

## 2.6  Nearest Neighbor

We now return to the issue how to represent $Q$. Since we are operating in real-valued spaces, some sort of function approximation method is called for. However, recall that $Q$ accepts a probability distribution (a sample set) as an input. This makes most existing function approximators (e.g., neural networks) inapplicable.

In our current implementation, nearest neighbor [11] is applied to represent $Q$. More specifically, our algorithm maintains a set of sample sets $\theta$ (belief states) annotated by an action $a$ and a $Q$-value $Q(\theta, a)$. When a new belief state $\theta'$ is encountered, its $Q$-value is obtained by finding the $k$ nearest neighbors in the database, and linearly averaging their $Q$-values. If there aren't sufficiently many neighbors (within a pre-specified maximum distance), $\theta'$ is added to the database; hence, the database grows over time.

Our approach uses KL divergence (relative entropy) as a distance function[1]. Technically, the KL-divergence between two continuous distributions is well-defined. When applied to sample sets, however, it cannot be computed. Hence, when evaluating the distance between two different sample sets, our approach maps them into continuous-valued densities using Gaussian kernels, and uses Monte Carlo sampling to approximate the KL divergence between them. This algorithm is fairly generic an extension of nearest neighbors to function approximation in density space, where densities are represented by samples. Space limitations preclude us from providing further detail (see [11, 12]).

# 3  Experimental Results

Preliminary results have been obtained in a world shown in two domains, one synthetic and one using a simulator of a RWI B21 robot.

In the synthetic environment (Figure 2a), the agents starts at the lower left corner. Its objective is to reach "heaven" which is either at the upper left corner or the lower right

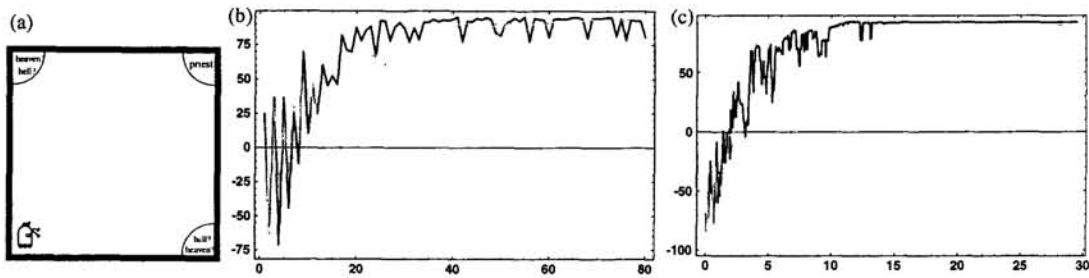

**Figure 2**: (a) The environment, schematically. (b) Average performance (reward) as a function of training episodes. The black graph corresponds to the smaller environment (25 steps min), the grey graph to the larger environment (50 steps min). (c) Same results, plotted as a function of number of backups (in thousands).

corner. The opposite location is "hell." The agent does not know the location of heaven, but it can ask a "priest" who is located in the upper right corner. Thus, an optimal solution requires the agent to go first to the priest, and then head to heaven. The state space contains a real-valued (coordinates of the agent) and discrete (location of heaven) component. Both are unobservable: In addition to not knowing the location of heaven, the agent also cannot sense its (real-valued) coordinates. 5% random motion noise is injected at each move. When an agent hits a boundary, it is penalized, but it is also told which boundary it hit (which makes it possible to infer its coordinates along one axis). However, notice that the *initial* coordinates of the agent are known.

The optimal solution takes approximately 25 steps; thus, a successful POMDP planner must be capable of looking 25 steps ahead. We will use the term "successful policy" to refer to a policy that always leads to heaven, even if the path is suboptimal. For a policy to be successful, the agent must have learned to first move to the priest (information gathering), and then proceed to the right target location.

Figures 2b&c show performance results, averaged over 13 experiments. The solid (black) curve in both diagrams plots the average cumulative reward $J$ as a function of the number of training episodes (Figure 2b), and as a function of the number of backups (Figure 2c). A successful policy was consistently found after 17 episodes (or 6,150 backups), in all 13 experiments. In our current implementation, 6,150 backups require approximately 29 minutes on a Pentium PC. In some experiments, a successful policy was identified in 6 episodes (less than 1,500 backups or 7 minutes). After a successful policy is found, further learning gradually optimizes the path. To investigate scaling, we doubled the size of the environment (quadrupling the size of the state space), making the optimal solution 50 steps long. The results are depicted by the gray curves in Figures 2b&c. Here a successful policy is consistently found after 33 episodes (10,250 backups, 58 minutes). In some runs, a successful policy is identified after only 14 episodes.

We also applied MC-POMDPs to a robotic *locate-and-retrieve task*. Here a robot (Figure 3a) is to find and grasp an object somewhere in its vicinity (at floor *or* table height). The robot's task is to grasp the object using its gripper. It is rewarded for successfully grasping the object, and penalized for unsuccessful grasps or for moving too far away from the object. The state space is continuous in $x$ and $y$ coordinates, and discrete in the object's height.

The robot uses a mono-camera system for object detection; hence, viewing the object from a single location is insufficient for its 3D localization. Moreover, initially the object might not be in sight of the robot's camera, so that the robot must look around first. In our simulation, we assume 30% general detection error (false-positive and false-negative), with additional Gaussian noise if the object is detected correctly. The robot's actions include turns (by a variable angle), translations (by a variable distance), and grasps (at one of two legal heights). Robot control is erroneous with a variance of 20% (in $x$-$y$-space) and 5% (in rotational space). Typical belief states range from uniformly distributed sample sets (initial belief) to samples narrowly focused on a specific $x$-$y$-$z$ location.

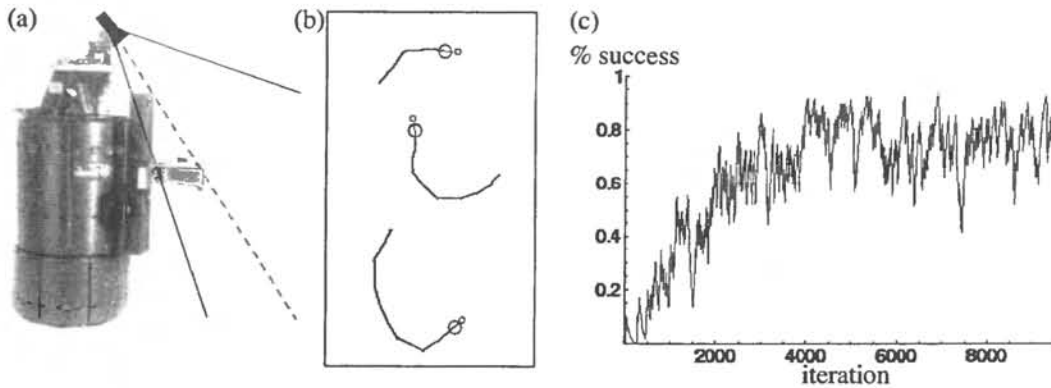

**Figure 3**: Find and fetch task: (a) The mobile robot with gripper and camera, holding the target object (experiments are carried out in simulation!), (b) three successful runs (trajectory projected into 2D), and (c) success rate as a function of number of planning steps.

Figure 3c shows the rate of successful grasps as a function of iterations (actions). While initially, the robot fails to grasp the object, after approximately 4,000 iterations its performance surpasses 80%. Here the planning time is in the order of 2 hours. However, the robot fails to reach 100%. This is in part because certain initial configurations make it impossible to succeed (e.g., when the object is too close to the maximum allowed distance), in part because the robot occasionally misses the object by a few centimeters. Figure 3b depicts three successful example trajectories. In all three, the robot initially searches the object, then moves towards it and grasps it successfully.

## 4   Discussion

We have presented a Monte Carlo approach for learning how to act in partially observable Markov decision processes (POMDPs). Our approach represents all belief distributions using samples drawn from these distributions. Reinforcement learning in belief space is applied to learn optimal policies, using a sample-based version of nearest neighbor for generalization. Backups are performed using Monte Carlo sampling. Initial experimental results demonstrate that our approach is applicable to real-valued domains, and that it yields good performance results in environments that are—by POMDP standards—relatively large.

## Footnotes

[1]Strictly speaking, KL divergence is not a distance metric, but this is ignored here.

## References

[1] AAAI Fall symposium on POMDPs. 1998. See `http://www.cs.duke.edu/~mlittman/talks/ pomdp-symposium.html`

[2] R. E. Bellman. *Dynamic Programming*. Princeton University Press, 1957.

[3] P. Dayan and T. J. Sejnowski. TD($\lambda$) converges with probability 1. 1993.

[4] D. Fox, W. Burgard, F. Dellaert, and S. Thrun. Monte carlo localization: Efficient position estimation for mobile robots. AAAI-99.

[5] M. Isard and A. Blake. Condensation: conditional density propagation for visual tracking. *International Journal of Computer Vision*, 1998.

[6] L.P. Kaelbling, M.L. Littman, and A.R. Cassandra. Planning and acting in partially observable stochastic domains. Submitted for publication, 1997.

[7] L.P. Kaelbling, M.L. Littman, and A.W. Moore. Reinforcement learning: A survey. *JAIR*, 4, 1996.

[8] K. Kanazawa, D. Koller, and S.J. Russell. Stochastic simulation algorithms for dynamic probabilistic networks. UAI-95.

[9] L.-J. Lin. Self-improving reactive agents based on reinforcement learning, planning and teaching. *Machine Learning*, 8, 1992.

[10] M.L. Littman, A.R. Cassandra, and L.P. Kaelbling. Learning policies for partially observable environments: Scaling up. ICML-95.

[11] A.W. Moore, C.G. Atkeson, and S.A. Schaal. Locally weighted learning for control. *AI Review*, 11, 1997.

[12] D. Ormoneit and S. Sen. Kernel-based reinforcement learning. TR 1999-8, Statistics, Stanford University, 1999.

[13] M. Pitt and N. Shephard. Filtering via simulation: auxiliary particle filter. *Journal of the American Statistical Association*, 1999.

[14] E. Sondik. *The Optimal Control of Partially Observable Markov Processes*. PhD thesis, Stanford, 1971.

[15] R.S. Sutton and A.G. Barto. *Reinforcement Learning: An Introduction*. MIT Press, 1998.

[16] M.A. Tanner. *Tools for Statistical Inference*. Springer Verlag, 1993.

[17] C. J. C. H. Watkins. *Learning from Delayed Rewards*. PhD thesis, King's College, Cambridge, 1989.